# Estimation of Intrinsic Dimensionality Using High-Rate Vector Quantization

**Maxim Raginsky and Svetlana Lazebnik**
Beckman Institute, University of Illinois
405 N Mathews Ave, Urbana, IL 61801
{maxim,slazebni}@uiuc.edu

## Abstract

We introduce a technique for dimensionality estimation based on the notion of *quantization dimension*, which connects the asymptotic optimal quantization error for a probability distribution on a manifold to its intrinsic dimension. The definition of quantization dimension yields a family of estimation algorithms, whose limiting case is equivalent to a recent method based on packing numbers. Using the formalism of high-rate vector quantization, we address issues of statistical consistency and analyze the behavior of our scheme in the presence of noise.

## 1. Introduction

The goal of *nonlinear dimensionality reduction* (NLDR) [1, 2, 3] is to find low-dimensional manifold descriptions of high-dimensional data. Most NLDR schemes require a good estimate of the intrinsic dimensionality of the data to be available in advance. A number of existing methods for estimating the intrinsic dimension (e.g., [3, 4, 5]) rely on the fact that, for data uniformly distributed on a $d$-dimensional compact smooth submanifold of $\mathbb{R}^D$, the probability of a small ball of radius $\epsilon$ around any point on the manifold is $\Theta(\epsilon^d)$. In this paper, we connect this argument with the notion of *quantization dimension* [6, 7], which relates the intrinsic dimension of a manifold (a *topological* property) to the asymptotic optimal quantization error for distributions on the manifold (an *operational* property). Quantization dimension was originally introduced as a theoretical tool for studying "non-standard" signals, such as singular distributions [6] or fractals [7]. However, to the best of our knowledge, it has not been previously used for dimension estimation in manifold learning. The definition of quantization dimension leads to a family of dimensionality estimation algorithms, parametrized by the *distortion exponent* $r \in [1, \infty)$, yielding in the limit of $r = \infty$ a scheme equivalent to Kégl's recent technique based on packing numbers [4].

To date, many theoretical aspects of intrinsic dimensionality estimation remain poorly understood. For instance, while the estimator bias and variance are assessed either heuristically [4] or exactly [5], scant attention is paid to robustness of each particular scheme against noise. Moreover, existing schemes do not fully utilize the potential for statistical consistency afforded by ergodicity of i.i.d. data: they compute the dimensionality estimate from a fixed training sequence (typically, the entire dataset of interest), whereas we show that an independent *test sequence* is necessary to avoid overfitting. In addition, using the framework of high-rate vector quantization allows us to analyze the performance of our scheme in the presence of noise.

## 2. Quantization-based estimation of intrinsic dimension

Let us begin by introducing the definitions and notation used in the rest of the paper. A *D-dimensional k-point vector quantizer* [6] is a measurable map $Q_k : \mathbb{R}^D \to \mathcal{C}$, where $\mathcal{C} = \{y_1, \ldots, y_k\} \subset \mathbb{R}^D$ is called the *codebook* and the $y_i$'s are called the *codevectors*. The number $\log_2 k$ is called the *rate* of the quantizer, in bits per vector. The sets $R_i \triangleq \{x \in \mathbb{R}^D : Q_k(x) = y_i\}, 1 \le i \le k$, are called the *quantizer cells* (or *partition regions*). The quantizer performance on a random vector $X$ distributed according to a probability distribution $\mu$ (denoted $X \sim \mu$) is measured by the *average rth-power distortion* $\delta_r(Q_k|\mu) \triangleq \mathrm{E}_\mu \|X - Q_k(X)\|^r$, $r \in [1, \infty)$, where $\|\cdot\|$ is the Euclidean norm on $\mathbb{R}^D$. In the sequel, we will often find it more convenient to work with the *quantizer error* $e_r(Q_k|\mu) \triangleq \delta_r(Q_k|\mu)^{1/r}$. Let $\mathcal{Q}_k$ denote the set of all $D$-dimensional $k$-point quantizers. Then the performance achieved by an *optimal k-point quantizer* on $X$ is $\delta_r^*(k|\mu) \triangleq \inf_{Q_k \in \mathcal{Q}_k} \delta_r(Q_k|\mu)$ or equivalently, $e_r^*(k|\mu) \triangleq \delta_r^*(k|\mu)^{1/r}$.

### 2.1. Quantization dimension

The dimensionality estimation method presented in this paper exploits the connection between the intrinsic dimension $d$ of a smooth compact manifold $M \subset \mathbb{R}^D$ (from now on, simply referred to as "manifold") and the asymptotic optimal quantization error for a regular probability distribution[1] on $M$. When the quantizer rate is high, the partition cells can be well approximated by $D$-dimensional balls around the codevectors. Then the regularity of $\mu$ ensures that the probability of such a ball of radius $\epsilon$ is $\Theta(\epsilon^d)$, and it can be shown [7, 6] that $e_r^*(k|\mu) = \Theta(k^{-1/d})$. This is referred to as the *high-rate* (or *high-resolution*) *approximation*, and motivates the definition of *quantization dimension* of order $r$:

$$d_r(\mu) \triangleq -\lim_{k \to \infty} \frac{\log k}{\log e_r^*(k|\mu)}.$$

The theory of high-rate quantization confirms that, for a regular $\mu$ supported on the manifold $M$, $d_r(\mu)$ exists for all $1 \le r \le \infty$ and equals the intrinsic dimension of $M$ [7, 6]. (The $r = \infty$ limit will be treated in Sec. 2.2.)

This definition immediately suggests an empirical procedure for estimating the intrinsic dimension of a manifold from a set of samples. Let $X^n = (X_1, \ldots, X_n)$ be $n$ i.i.d. samples from an unknown regular distribution $\mu$ on the manifold. We also fix some $r \in [1, \infty)$. Briefly, we select a range $k_1 \le k \le k_2$ of codebook sizes for which the high-rate approximation holds (see Sec. 3 for implementation details), and design a sequence of quantizers $\{\hat{Q}_k\}_{k=k_1}^{k_2}$ that give us good approximations $\hat{e}_r(k|\mu)$ to the optimal error $e_r^*(k|\mu)$ over the chosen range of $k$. Then an estimate of the intrinsic dimension is obtained by plotting $\log k$ vs. $-\log \hat{e}_r(k|\mu)$ and measuring the slope of the plot over the chosen range of $k$ (because the high-rate approximation holds, the plot is linear).

This method hinges on estimating reliably the optimal errors $e_r^*(k|\mu)$. Let us explain how this can be achieved. The ideal quantizer for each $k$ should minimize the *training error*

$$e_r(Q_k|\mu_{\text{train}}) = \left( \frac{1}{n} \sum_{i=1}^n \|X_i - Q_k(X_i)\|^r \right)^{1/r},$$

where $\mu_{\text{train}}$ is the corresponding empirical distribution. However, finding this *empirically optimal* quantizer is, in general, an intractable problem, so in practice we merely strive to produce a quantizer $\hat{Q}_k$ whose error $e_r(\hat{Q}_k|\mu_{\text{train}})$ is a good approximation to the *minimal empirical error* $e_r^*(k|\mu_{\text{train}}) \triangleq \inf_{Q_k \in \mathcal{Q}_k} e_r(Q_k|\mu_{\text{train}})$ (the issue of quantizer design is discussed in Sec. 3). However, while minimizing the training error is necessary for obtaining a statistically consistent approximation to an optimal quantizer for $\mu$, the training error itself is an optimistically biased estimate of $e_r^*(k|\mu)$ [8]: intuitively, this is due to the fact that an empirically designed quantizer overfits the training set. A less biased estimate is given by the performance of $\hat{Q}_k$ on a *test sequence* independent from the training set. Let $Z^m = (Z_1, \ldots, Z_m)$ be $m$ i.i.d. samples from $\mu$, independent from $X^n$. Provided $m$ is sufficiently large, the law of large numbers guarantees that the empirical average

$$e_r(\hat{Q}_k|\mu_{\text{test}}) = \left( \frac{1}{m} \sum_{i=1}^{m} \|Z_i - \hat{Q}_k(Z_i)\|^r \right)^{1/r}$$

will be a good estimate of the *test error* $e_r(\hat{Q}_k|\mu)$. Using learning-theoretic formalism [8], one can show that the test error of an empirically optimal quantizer is a *strongly consistent* estimate of $e_r^*(k|\mu)$, i.e., it converges almost surely to $e_r^*(k|\mu)$ as $n \to \infty$. Thus, we take $\hat{e}_r(k|\mu) = e_r(\hat{Q}_k|\mu_{\text{test}})$. In practice, therefore, the proposed scheme is statistically consistent to the extent that $\hat{Q}_k$ is close to the optimum.

## 2.2. The $r = \infty$ limit and packing numbers

If the support of $\mu$ is compact (which is the case with all probability distributions considered in this paper), then the limit $e_\infty(Q_k|\mu) = \lim_{r \to \infty} e_r(Q_k|\mu)$ exists and gives the "worst-case" quantization error of $X$ by $Q_k$:

$$e_\infty(Q_k|\mu) = \max_{x \in \text{supp}(\mu)} \|x - Q_k(x)\|.$$

The optimum $e_\infty^*(k|\mu) = \inf_{Q_k \in \mathcal{Q}_k} e_\infty(Q_k|\mu)$ has an interesting interpretation as the smallest covering radius of the most parsimonious covering of $\text{supp}(\mu)$ by $k$ or fewer balls of equal radii [6]. Let us describe how the $r = \infty$ case is equivalent to dimensionality estimation using packing numbers [4]. The *covering number* $N_M(\epsilon)$ of a manifold $M \subset \mathbb{R}^D$ is defined as the size of the smallest covering of $M$ by balls of radius $\epsilon > 0$, while the *packing number* $P_M(\epsilon)$ is the cardinality of the maximal set $S \subset M$ with $\|x - y\| \geq \epsilon$ for all distinct $x, y \in S$. If $d$ is the dimension of $M$, then $N_M(\epsilon) = \Theta(\epsilon^{-d})$ for small enough $\epsilon$, leading to the definition of the *capacity dimension*: $d_{\text{cap}}(M) \triangleq -\lim_{\epsilon \to 0} \frac{\log N_M(\epsilon)}{\log \epsilon}$. If this limit exists, then it equals the intrinsic dimension of $M$. Alternatively, Kégl [4] suggests using the easily proved inequality $N_M(\epsilon) \leq P_M(\epsilon) \leq N_M(\epsilon/2)$ to express the capacity dimension in terms of packing numbers as $d_{\text{cap}}(M) = -\lim_{\epsilon \to 0} \frac{\log P_M(\epsilon)}{\log \epsilon}$.

Now, a simple geometric argument shows that, for any $\mu$ supported on $M$, $P_M(e_\infty^*(k|\mu)) > k$ [6]. On the other hand, $N_M(e_\infty^*(k|\mu)) \leq k$, which implies that $P_M(2e_\infty^*(k|\mu)) \leq k$. Let $\{\epsilon_k\}$ be a sequence of positive reals converging to zero, such that $\epsilon_k = e_\infty^*(k|\mu)$. Let $k_0$ be such that $\log \epsilon_k < 0$ for all $k \geq k_0$. Then it is not hard to show that

$$-\frac{\log P_M(2\epsilon_k)}{\log 2\epsilon_k - 1} \leq -\frac{\log k}{\log e_\infty^*(k|\mu)} < -\frac{\log P_M(\epsilon_k)}{\log \epsilon_k}, \qquad k \geq k_0.$$

In other words, there exists a decreasing sequence $\{\epsilon_k\}$, such that for sufficiently large values of $k$ (i.e., in the high-rate regime) the ratio $-\log k / \log e_\infty^*(k|\mu)$ can be approximated increasingly finely both from below and from above by quantities involving the packing numbers $P_M(\epsilon_k)$ and $P_M(2\epsilon_k)$ and converging to the common value $d_{\text{cap}}(M)$.

This demonstrates that the $r = \infty$ case of our scheme is numerically equivalent to Kégl's method based on packing numbers.

For a finite training set, the $r = \infty$ case requires us to find an empirically optimal $k$-point quantizer w.r.t. the worst-case $\ell_2$ error — a task that is much more computationally complex than for the $r = 2$ case (see Sec. 3 for details). In addition to computational efficiency, other important practical considerations include sensitivity to sampling density and noise. In theory, this worst-case quantizer is completely insensitive to variations in sampling density, since the optimal error $e_\infty^*(k|\mu)$ is the same for all $\mu$ with the same support. However, this advantage is offset in practice by the increased sensitivity of the $r = \infty$ scheme to noise, as explained next.

### 2.3. Estimation with noisy data

Random noise transforms "clean" data distributed according to $\mu$ into "noisy" data distributed according to some other distribution $\nu$. This will cause the empirically designed quantizer to be matched to the noisy distribution $\nu$, whereas our aim is to estimate optimal quantizer performance on the original clean data. To do this, we make use of the $r$th-order *Wasserstein distance* [6] between $\mu$ and $\nu$, defined as $\bar{\rho}_r(\mu, \nu) \triangleq \inf_{X \sim \mu, Y \sim \nu} (\mathrm{E} \|X - Y\|^r)^{1/r}$, $r \in [1, \infty)$, where the infimum is taken over all pairs $(X, Y)$ of jointly distributed random variables with the respective marginals $\mu$ and $\nu$. It is a natural measure of *quantizer mismatch*, i.e., the difference in performance that results from using a quantizer matched to $\nu$ on data distributed according to $\mu$ [9]. Let $\nu_n$ denote the empirical distribution of $n$ i.i.d. samples of $\nu$. It is possible to show (details omitted for lack of space) that for an empirically optimal $k$-point quantizer $Q_{k,r}^*$ trained on $n$ samples of $\nu$, $|e_r(Q_{k,r}^*|\nu) - e_r^*(k|\mu)| \leq 2\bar{\rho}_r(\nu_n, \nu) + \bar{\rho}_r(\mu, \nu)$. Moreover, $\nu_n$ converges to $\nu$ in the Wasserstein sense [6]: $\lim_{n \to \infty} \bar{\rho}_r(\nu_n, \nu) = 0$. Thus, provided the training set is sufficiently large, the distortion estimation error is controlled by $\bar{\rho}_r(\mu, \nu)$.

Consider the case of isotropic additive Gaussian noise. Let $W$ be a $D$-dimensional zero-mean Gaussian with covariance matrix $K = \sigma^2 I_D$, where $I_D$ is the $D \times D$ identity matrix. The noisy data are described by the random variable $X + W = Y \sim \nu$, and

$$\bar{\rho}_r(\mu, \nu) \leq \sqrt{2}\sigma \left[ \frac{\Gamma((r + D)/2)}{\Gamma(D/2)} \right]^{1/r},$$

where $\Gamma$ is the gamma function. In particular, $\bar{\rho}_2(\mu, \nu) \leq \sigma\sqrt{D}$. The magnitude of the bound, and hence the worst-case sensitivity of the estimation procedure to noise, is controlled by the noise variance, by the extrinsic dimension, and by the distortion exponent. The factor involving the gamma functions grows without bound both as $D \to \infty$ and as $r \to \infty$, which suggests that the susceptibility of our algorithm to noise increases with the extrinsic dimension of the data and with the distortion exponent.

## 3. Experimental results

We have evaluated our quantization-based scheme for two choices of the distortion exponent, $r = 2$ and $r = \infty$. For $r = 2$, we used the $k$-means algorithm to design the quantizers. For $r = \infty$, we have implemented a Lloyd-type algorithm, which alternates two steps: (1) the *minimum-distortion encoder*, where each sample $X_i$ is mapped to its nearest neighbor in the current codebook, and (2) the *centroid decoder*, where the center of each region is recomputed as the center of the minimum enclosing ball of the samples assigned to that region. It is clear that the decoder step locally minimizes the worst-case error (the largest distance of any sample from the center). Using a simple randomized algorithm, the minimum enclosing ball can be found in $O((D + 1)!(D + 1)N)$ time, where $N$ is the number of samples in the region [10]. Because of this dependence on $D$, the running time of the Lloyd algorithm becomes prohibitive in high dimensions, and even for $D < 10$ it is an

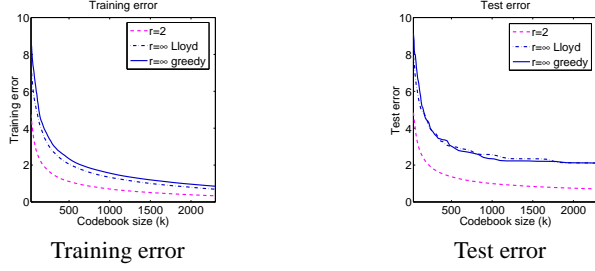

Training error          Test error

Figure 1: Training and test error vs. codebook size on the swiss roll (Figure 2 (a)). Dashed line: $r = 2$ ($k$-means), dash-dot: $r = \infty$ (Lloyd-type), solid: $r = \infty$ (greedy).

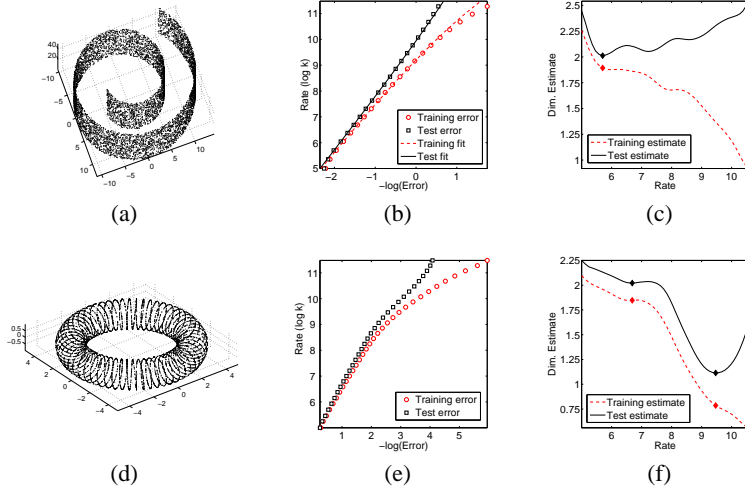

(a)          (b)          (c)

(d)          (e)          (f)

Figure 2: (a) The swiss roll (20,000 samples). (b) Plot of rate vs. negative log of the quantizer error (log-log curves), together with parametric curves fitted using linear least squares (see text). (c) Slope (dimension) estimates: 1.88 (training) and 2.04 (test). (d) Toroidal spiral (20,000 samples). (e) Log-log curves, exhibiting two distinct linear parts. (f) Dimension estimates: 1.04 (training), 2.02 (test) in the low-rate region, 0.79 (training), 1.11 (test) in the high-rate region.

order of magnitude slower than $k$-means. Thus, we were compelled to also implement a greedy algorithm reminiscent of Kégl's algorithm for estimating the packing number [4]: supposing that $k-1$ codevectors have already been selected, the $k$th one is chosen to be the sample point with the largest distance from the nearest codevector. Because this is the point that gives the worst-case error for codebook size $k-1$, adding it to the codebook lowers the error. We generate several codebooks, initialized with different random samples, and then choose the one with the smallest error. For the experiment shown in Figure 3, the training error curves produced by this greedy algorithm were on average $21\%$ higher than those of the Lloyd algorithm, but the test curves were only $8\%$ higher. In many cases, the two test curves are visually almost coincident (Figure 1). Therefore, in the sequel, we report only the results for the greedy algorithm for the $r = \infty$ case.

Our first synthetic dataset (Fig. 2 (a)) is the 2D "swiss roll" embedded in $\mathbb{R}^3$ [2]. We split the samples into 4 equal parts and use each part in turn for training and the rest for testing. This cross-validation setup produces four sets of error curves, which we average to obtain an improved estimate. We sample quantizer rates in increments of $0.1$ bits. The lowest rate is 5 bits, and the highest rate is chosen as $\log(n/2)$, where $n$ is the size of the training set.

The high-rate approximation suggests the asymptotic form $\Theta(k^{-1/d})$ for the quantizer error

as a function of codebook size $k$. To validate this approximation, we use linear least squares to fit curves of the form $a + b\,k^{-1/2}$ to the $r = 2$ training and test distortion curves for the the swiss roll. The fitting procedure yields estimates of $-0.22 + 29.70k^{-1/2}$ and $0.10 + 28.41k^{-1/2}$ for the training and test curves, respectively. These estimates fit the observed data well, as shown in Fig. 2(b), a plot of rate vs. the negative logarithm of the training and test error ("log-log curves" in the following). Note that the additive constant for the training error is negative, reflecting the fact that the training error of the empirical quantizer is identically zero when $n = k$ (each sample becomes a codevector). On the other hand, the test error has a positive additive constant as a consequence of quantizer suboptimality. Significantly, the fit deteriorates as $n/k \to 1$, as the average number of training samples per quantizer cell becomes too small to sustain the exponentially slow decay required for the high-rate approximation.

Fig. 2(c) shows the slopes of the training and test log-log curves, obtained by fitting a line to each successive set of 10 points. These slopes are, in effect, rate-dependent dimensionality estimates for the dataset. Note that the training slope is always below the test slope; this is a consequence of the "optimism" of the training error and the "pessimism" of the test error (as reflected in the additive constants of the parametric fits). The shapes of the two slope curves are typical of many "well-behaved" datasets. At low rates, both the training and the test slopes are close to the extrinsic dimension, reflecting the global geometry of the dataset. As rate increases, the local manifold structure is revealed, and the slope yields its intrinsic dimension. However, as $n/k \to 1$, the quantizer begins to "see" isolated samples instead of the manifold structure. Thus, the training slope begins to fall to zero, and the test slope rises, reflecting the failure of the quantizer to generalize to the test set. For most datasets in our experiments, a good intrinsic dimensionality estimate is given by the first minimum of the test slope where the line-fitting residual is sufficiently low (marked by a diamond in Fig. 2(c)). For completeness, we also report the slope of the training curve at the same rate (note that the training curve may not have local minima because of its tendency to fall as the rate increases). Interestingly, some datasets yield several well-defined dimensionality estimates at different rates. Fig. 2(d) shows a toroidal spiral embedded in $\mathrm{I\!R}^3$, which at larger scales "looks" like a torus, while at smaller scales the 1D curve structure becomes more apparent. Accordingly, the log-log plot of the test error (Fig. 2(e)) has two distinct linear parts, yielding dimension estimates of 2.02 and 1.11, respectively (Fig. 2(f)).

Recall from Sec. 2.1 that the high-rate approximation for regular probability distributions is based on the assumption that the intersection of each quantizer cell with the manifold is a $d$-dimensional neighborhood of that manifold. Because we compute our dimensionality estimate at a rate for which this approximation is valid, we know that the empirically optimal quantizer at this rate partitions the data into clusters that are locally $d$-dimensional. Thus, our dimensionality estimation procedure is also useful for finding a clustering of the data that respects the intrinsic neighborhood structure of the manifold from which it is sampled. As an expample, for the toroidal spiral of Fig. 2(c), we obtain two distinct dimensionality estimates of 2 and 1 at rates 6.6 and 9.4, respectively (Fig. 2(f)). Accordingly, quantizing the spiral at the lower (resp. higher) rate yields clusters that are locally two-dimensional (resp. one-dimensional).

To ascertain the effect of noise and extrinsic dimension on our method, we have embedded the swiss roll in dimensions 4 to 8 by zero-padding the coordinates and applying a random orthogonal matrix, and added isotropic zero-mean Gaussian noise in the high-dimensional space, with $\sigma = 0.2, 0.4, \ldots, 1$. First, we have verified that the $r = 2$ estimator behaves in agreement with the Wasserstein bound from Sec. 2.3. The top part of Fig. 3(a) shows the maximum differences between the noisy and the noiseless test error curves for each combination of $D$ and $\sigma$, and the bottom part shows the corresponding values of the Wasserstein bound $\sigma\sqrt{D}$ for comparison. For each value of $\sigma$, the test error of the empirically designed quantizer differs from the noiseless case by $O(\sqrt{D})$, while, for a fixed $D$, the difference

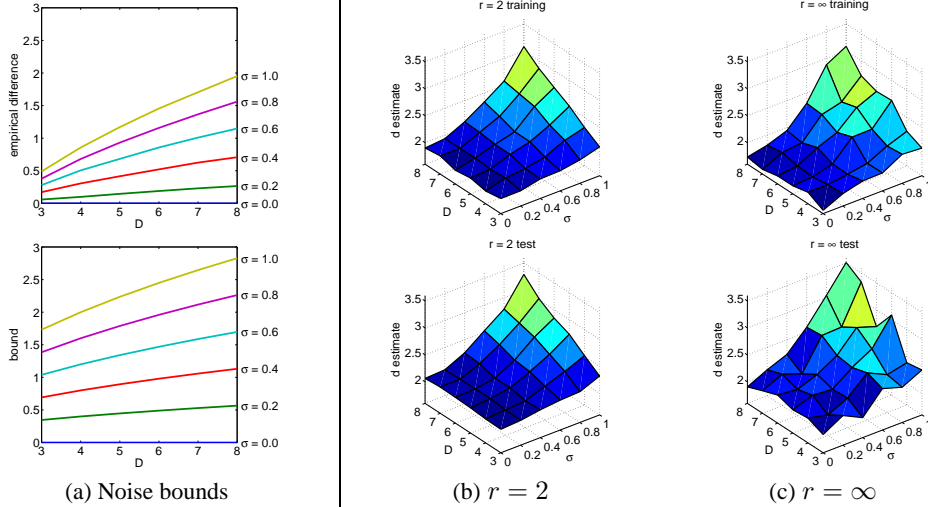

(a) Noise bounds      (b) $r = 2$      (c) $r = \infty$

Figure 3: (a) Top: empirically observed differences between noisy and noiseless test curves; bottom: theoretically derived bound $\left(\sigma\sqrt{D}\right)$. (b) Height plot of dimension estimates for the $r = 2$ algorithm as a function of $D$ and $\sigma$. Top: training estimates, bottom: test estimates. (c) Dimension estimates for $r = \infty$. Top: training, bottom: test. Note that the training estimates are consistently lower than the test estimates: the average difference is 0.17 (resp. 0.28) for the $r = 2$ (resp. $r = \infty$) case.

of the noisy and noiseless test errors grows as $O(\sigma)$. As predicted by the bound, the additive constant in the parametric form of the test error increases with $\sigma$, resulting in larger slopes of the log-log curve and therefore higher dimension estimates. This is reflected in Figs. 3(b) and (c), which show training and test dimensionality estimates for $r = 2$ and $r = \infty$, respectively. The $r = \infty$ estimates are much less stable than those for $r = 2$ because the $r = \infty$ (worst-case) error is controlled by outliers and often stays constant over a range of rates. The piecewise-constant shape of the test error curves (see Fig. 1) results in log-log plots with unstable slopes.

Table 1 shows a comparative evaluation on the MNIST handwritten digits database[2] and a face video.[3] The MNIST database contains 70,000 images at resolution $28 \times 28$ ($D = 784$), and the face video has 1965 frames at resolution $28 \times 20$ ($D = 560$). For each of the resulting 11 datasets (taking each digit separately), we used half the samples for training and half for testing. The first row of the table shows dimension estimates obtained using a baseline regression method [3]: for each sample point, a local estimate is given by the first local minimum of the curve $\frac{\mathrm{d}\log\ell}{\mathrm{d}\log\epsilon(\ell)}$, where $\epsilon(\ell)$ is the distance from the point to its $\ell$th nearest neighbor, and a global estimate is then obtained by averaging the local estimates. The rest of the table shows the estimates obtained from the training and test curves of the $r = 2$ quantizer and the (greedy) $r = \infty$ quantizer. Comparative examination of the results shows that the $r = \infty$ estimates tend to be fairly low, which is consistent with the experimental findings of Kégl [4]. By contrast, the $r = 2$ estimates seem to be most resistant to negative bias. The relatively high values of the dimension estimates reflect the many degrees of freedom found in handwritten digits, including different scale, slant and thickness of the strokes, as well as the presence of topological features (i.e., loops in 2's or extra horizontal bars in 7's). The lowest dimensionality is found for 1's, while the highest is found for 8's, reflecting the relative complexities of different digits. For the face dataset, the different dimensionality estimates range from 4.25 to 8.30. This dataset certainly contains enough degrees of freedom to justify such high estimates, including changes in pose

Table 1: Performance on the MNIST dataset and on the Frey faces dataset.

| | Handwritten digits (MNIST data set) | | | | | | | | | | Faces |
|---|---|---|---|---|---|---|---|---|---|---|---|
| | 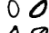 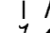 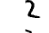 | 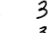  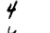 | 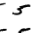 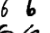 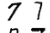 | 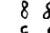  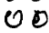 | 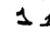 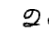 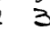 |  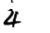 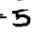 | 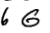 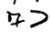 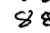 | | | |  |
| # samples | 6903 | 7877 | 6990 | 7141 | 6824 | 6313 | 6876 | 7293 | 6825 | 6958 | 1965 |
| Regression | 11.14 | 7.86 | 12.79 | 13.39 | 11.98 | 13.05 | 11.19 | 10.42 | 13.79 | 11.26 | 5.63 |
| $r = 2$ train | 12.39 | 6.51 | 16.04 | 15.38 | 13.22 | 14.63 | 12.05 | 12.32 | 19.80 | 13.44 | 5.70 |
| $r = 2$ test | 15.47 | 7.11 | 20.89 | 19.78 | 16.79 | 19.80 | 16.02 | 16.02 | 20.07 | 17.46 | 8.30 |
| $r = \infty$ train | 10.33 | 8.19 | 10.15 | 12.63 | 9.87 | 8.49 | 9.85 | 8.10 | 10.88 | 7.40 | 4.25 |
| $r = \infty$ test | 9.02 | 6.61 | 13.98 | 12.21 | 7.26 | 10.46 | 9.08 | 9.92 | 14.03 | 9.59 | 6.39 |

and facial expression, as well as camera jitter.[4] Finally, for both the digits and the faces, significant noise in the dataset additionally inflated the estimates.

## 4. Discussion

We have demonstrated an approach to intrinsic dimensionality estimation based on high-rate vector quantization. A crucial distinguishing feature of our method is the use of an independent test sequence to ensure statistical consistency and avoid underestimating the dimension. Many existing methods are well-known to exhibit a negative bias in high dimensions [4, 5]. This can have serious implications in practice, as it may result in low-dimensional representations that lose essential features of the data. Our results raise the possibility that this negative bias may be indicative of overfitting. In the future we plan to integrate our proposed method into a unified package of quantization-based algorithms for estimating the intrinsic dimension of the data, obtaining its dimension-reduced manifold representation, and compressing the low-dimensional data [11].

### Acknowledgments

Maxim Raginsky was supported by the Beckman Institute Postdoctoral Fellowship. Svetlana Lazebnik was partially supported by the National Science Foundation grants IIS-0308087 and IIS-0535152.

## Footnotes

[1] A probability distribution $\mu$ on $\mathbb{R}^D$ is *regular of dimension d* [6] if it has compact support and if there exist constants $c, \epsilon_0 > 0$, such that $c^{-1} \epsilon^d \le \mu(B(a, \epsilon)) \le c \epsilon^d$ for all $a \in \text{supp}(\mu)$ and all $\epsilon \in (0, \epsilon_0)$, where $B(a, \epsilon)$ is the open ball of radius $\epsilon$ centered at $a$. If $M \subset \mathbb{R}^D$ is a $d$-dimensional smooth compact manifold, then any $\mu$ with $M = \text{supp}(\mu)$ that possesses a smooth, strictly positive density w.r.t. the normalized surface measure on $M$ is regular of dimension $d$.

[2] http://yann.lecun.com/exdb/mnist/

[3] http://www.cs.toronto.edu/~roweis/data.html, B. Frey and S. Roweis.

[4]Interestingly, Brand [3] reports an intrinsic dimension estimate of 3 for this data set. However, he used only a 500-frame subsequence and introduced additional mirror symmetry.

## References

[1] S.T. Roweis and L.K. Saul. Nonlinear dimensionality reduction by locally linear embedding. *Science*, 290:2323–2326, December 2000.

[2] J.B. Tenenbaum, V. de Silva, and J.C. Langford. A global geometric framework for nonlinear dimensionality reduction. *Science*, 290:2319–2323, December 2000.

[3] M. Brand. Charting a manifold. In *NIPS 15*, pages 977–984, Cambridge, MA, 2003. MIT Press.

[4] B. Kégl. Intrinsic dimension estimation using packing numbers. In *NIPS 15*, volume 15, Cambridge, MA, 2003. MIT Press.

[5] E. Levina and P.J. Bickel. Maximum likelihood estimation of intrinsic dimension. In *NIPS 17*, Cambridge, MA, 2005. MIT Press.

[6] S. Graf and H. Luschgy. *Foundations of Quantization for Probability Distributions*. Springer-Verlag, Berlin, 2000.

[7] P.L. Zador. Asymptotic quantization error of continuous signals and the quantization dimension. *IEEE Trans. Inform. Theory*, IT-28:139–149, March 1982.

[8] T. Linder. Learning-theoretic methods in vector quantization. In L. Györfi, editor, *Principles of Nonparametric Learning*. Springer-Verlag, New York, 2001.

[9] R.M. Gray and L.D. Davisson. Quantizer mismatch. *IEEE Trans. Commun.*, 23:439–443, 1975.

[10] E. Welzl. Smallest enclosing disks (balls and ellipsoids). In *New Results and New Trends in Computer Science*, volume 555 of *LNCS*, pages 359–370. Springer, 1991.

[11] M. Raginsky. A complexity-regularized quantization approach to nonlinear dimensionality reduction. *Proc. 2005 IEEE Int. Symp. Inform. Theory*, pages 352–356.

